# Self-regulation Mechanism of Temporally Asymmetric Hebbian Plasticity

**Narihisa Matsumoto**
Graduate School of Science and Engineering
Saitama University:
RIKEN Brain Science Institute
Saitama 351-0198, Japan
*xmatumo@brain.riken.go.jp*

**Masato Okada**
RIKEN Brain Science Institute
Saitama 351-0198, Japan
*okada@brain.riken.go.jp*

## Abstract

Recent biological experimental findings have shown that the synaptic plasticity depends on the relative timing of the pre- and post-synaptic spikes which determines whether Long Term Potentiation (LTP) occurs or Long Term Depression (LTD) does. The synaptic plasticity has been called "Temporally Asymmetric Hebbian plasticity (TAH)". Many authors have numerically shown that spatio-temporal patterns can be stored in neural networks. However, the mathematical mechanism for storage of the spatio-temporal patterns is still unknown, especially the effects of LTD. In this paper, we employ a simple neural network model and show that interference of LTP and LTD disappears in a sparse coding scheme. On the other hand, it is known that the covariance learning is indispensable for storing sparse patterns. We also show that TAH qualitatively has the same effect as the covariance learning when spatio-temporal patterns are embedded in the network.

## 1 Introduction

Recent biological experimental findings have indicated that the synaptic plasticity depends on the relative timing of the pre- and post- synaptic spikes which determines whether Long Term Potentiation (LTP) occurs or Long Term Depression (LTD) does [1, 2, 3]. LTP occurs when a presynaptic firing precedes a postsynaptic one by no more than about 20ms. In contrast, LTD occurs when a presynaptic firing follows a postsynaptic one. A rapid transition occurs between LTP and LTD within a time difference of a few ms. Such a learning rule is called "Temporally Asymmetric Hebbian learning (TAH)" [4, 5] or "Spike Timing Dependent synaptic Plasticity (STDP)" [6]. Many authors have numerically shown that spatio-temporal patterns can be stored in neural networks [6, 7, 8, 9, 10, 11]. Song et al. discussed the variablity of spike generation about the network consisting of spiking neurons using TAH [6]. They found that the condition that the area of LTD was slightly larger than that of LTP was indispensable of the stability. Namely, the balance of LTP and LTD is crucial. Yoshioka also discussed the associative memory network

consisting of spiking neurons using TAH [11]. He found that the area of LTP was needed to be equal to that of LTD for stable retrieval. Munro and Hernandez numerically showed that a network can retrieve spatio-temporal patterns even in a noisy environment owing to LTD [9]. However, they did not discuss the reason why TAH was effective in terms of the storage and retrieval of the spatio-temporal patterns. Since TAH has not only the effect of LTP but that of LTD, the interference of LTP and LTD may prevent retrieval of the patterns. To investigate this unknown mathematical mechanism for retrieval, we employ an associative memory network consisting of binary neurons. To simplify the dynamics of internal potential enables us to analyze the details of the retrieval process. We use a learning rule that is the similar formulation in the previous works. We show the mechanism that the spatio-temporal patterns can be retrieved in this network.

There are many works concerned with associative memory networks that store spatio-temporal patterns by the covariance learning [12, 13]. Many biological findings imply that sparse coding schemes may be used in the brain [14]. It is well-known that the covariance learning is indispensable when the sparse patterns are embedded in a network as attractors [15, 16]. The information on the firing rate for the stored patterns is *not* indispensable for TAH, although it is indispensable for the covariance learning. We theoretically show that TAH qualitatively has the same effect as the covariance learning when the spatio-temporal patterns are embedded in the network. This means that the difference in spike times induces LTP or LTD, and the effect of the firing rate information can be canceled out by this spike time difference. We conclude that this is the reason why TAH doesn't require the information on the firing rate for the stored patterns.

## 2 Model

We investigate a network consisting of $N$ binary neurons that are connected mutually. In this paper, we consider the case of $N \rightarrow \infty$. We use a neuronal model with binary state, $\{0, 1\}$. We also use discrete time steps and the following synchronous updating rule,

$$u_i(t) = \sum_{j=1}^{N} J_{ij} x_j(t), \tag{1}$$

$$x_i(t+1) = \Theta(u_i(t) - \theta), \tag{2}$$

$$\Theta(u) = \begin{cases} 1, & u \geq 0 \\ 0, & u < 0, \end{cases} \tag{3}$$

where $x_i(t)$ is the state of the $i$-th neuron at time $t$, $u_i(t)$ its internal potential, and $\theta$ a uniform threshold. If the $i$-th neuron fires at time $t$, its state is $x_i(t) = 1$; otherwise, $x_i(t) = 0$. The specific value of the threshold is discussed later. $J_{ij}$ is the synaptic weight from the $j$-th neuron to the $i$-th neuron. Each element $\xi_i^\mu$ of the $\mu$-th memory pattern $\boldsymbol{\xi}^\mu = (\xi_1^\mu, \xi_2^\mu, \cdots, \xi_N^\mu)$ is generated independently by,

$$\text{Prob}[\xi_i^\mu = 1] = 1 - \text{Prob}[\xi_i^\mu = 0] = f. \tag{4}$$

The expectation of $\boldsymbol{\xi}^\mu$ is $\text{E}[\xi_i^\mu] = f$, and thus, $f$ can be considered as the mean firing rate of the memory pattern. The memory pattern is "sparse" when $f \rightarrow 0$, and this coding scheme is called "sparse coding". The synaptic weight $J_{ij}$ follows the synaptic plasticity that depends on the difference in spike times between the $i$-th (post-) and $j$-th (pre-) neurons. The difference determines whether LTP occurs or LTD does. Such a learning rule is called "Temporally Asymmetric Hebbian learning (TAH)" or "Spike Timing Dependent synaptic Plasticity (STDP)". This biological

experimental finding indicates that LTP or LTD is induced when the difference in the pre- and post-synaptic spike times falls within about 20ms [3] (Figure 1(a)). We define that one time step in equations (1)–(3) corresponds to 20ms in Figure 1(a), and a time duration within 20ms is ignored (Figure 1(b)). Figure 1(b) shows that LTP occurs when the $j$-th neuron fires one time step before the $i$-th neuron does, $\xi_i^{\mu+1} = \xi_j^{\mu} = 1$, and that LTD occurs when the $j$-th neuron fires one time step after the $i$-th neuron does, $\xi_i^{\mu-1} = \xi_j^{\mu} = 1$. The previous work indicates the blance of LTP and LTD is significant [6]. Therefore, we define that the area of LTP is the

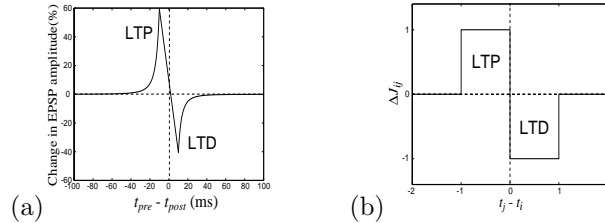

(a)    (b)

Figure 1: Temporally Asymmetric Hebbian plasticity. (a): The result of biological finding [3] and (b): the learning rule in our model. LTP occurs when the $j$-th neuron fires one time step before the $i$-th one. On the contrary, LTD occurs when the $j$-th neuron fires one time step after the $i$-th one. Synaptic weight $J_{ij}$ is followed by this rule.

same as that of LTD, and that the amplitude of LTP is also the same as that of LTD. On the basis of these definitions, we employ the following learning rule,

$$J_{ij} = \frac{1}{Nf(1-f)} \sum_{\mu=1}^{p} (\xi_i^{\mu+1}\xi_j^{\mu} - \xi_i^{\mu-1}\xi_j^{\mu}). \qquad (5)$$

The number of memory patterns is $p = \alpha N$ where $\alpha$ is defined as the "loading rate". There is a critical value $\alpha_C$ of loading rate. If the loading rate is larger than $\alpha_C$, the pattern sequence becomes unstable. $\alpha_C$ is called the "storage capacity". The previous works have shown that the learning method of equation (5) can store spatio-temporal patterns, that is, pattern sequences [9, 10]. We show that $p$ memory patterns are retrieved periodically like $\boldsymbol{\xi}^1 \rightarrow \boldsymbol{\xi}^2 \rightarrow \cdots \rightarrow \boldsymbol{\xi}^p \rightarrow \boldsymbol{\xi}^1 \rightarrow \cdots$. In other words, $\boldsymbol{\xi}^1$ is retrieved at $t = 1$, $\boldsymbol{\xi}^2$ at $t = 2$, and $\boldsymbol{\xi}^1$ at $t = p + 1$.

Here, we discuss the value of threshold $\theta$. It is well-known that the threshold value should be controlled according to the progress of the retrieval process time-dependently [15, 16]. One candidate algorithm for controlling the threshold value is to maintain the mean firing rate of the network at that of memory pattern, $f$, as follows,

$$f = \frac{1}{N}\sum_{i=1}^{N} x_i(t) = \frac{1}{N}\sum_{i=1}^{N} \Theta(u_i(t) - \theta(t)). \qquad (6)$$

It is known that the obtained threshold value is nearly optimal, since it approximately gives a maximal storage capacity value [16].

## 3   Theory

Many neural network models that store and retrieve sequential patterns by TAH have been discussed by many authors [7, 8, 9, 10]. They have numerically shown that

TAH is effective for storing pattern sequences. For example, Munro and Hernandez showed that their model could retrieve a stored pattern sequence even in a noisy environment [9]. However, the previous works have not mentioned the reason why TAH is effective. Exploring such a mechanism is the main purpose of our paper.

Here, we discuss the mechanism that the network learned by TAH can store and retrieve sequential patterns. Before providing details of the retrieval process, we discuss a simple situation where the number of memory patterns is very small relative to the number of neurons, i.e., $p \sim O(1)$. Let the state at time $t$ be the same as the $t$-th memory pattern: $\boldsymbol{x}(t) = \boldsymbol{\xi}^t$. Then, the internal potential $u_i(t)$ of the equation (1) is given by,

$$u_i(t) = \xi_i^{t+1} - \xi_i^{t-1}. \tag{7}$$

$u_i(t)$ depends on two independent random variables, $\xi_i^{t+1}$ and $\xi_i^{t-1}$, according to the equation (4). The first term $\xi_i^{t+1}$ of the equation (7) is a signal term for the recall of the pattern $\boldsymbol{\xi}^{t+1}$, which is designed to be retrieved at time $t+1$, and the second term $\xi_i^{t-1}$ can interfere in retrieval of $\boldsymbol{\xi}^{t+1}$. According to the equation (7), $u_i(t)$ takes a value of 0, $-1$ or $+1$. $\xi_i^{t-1} = 1$ means that the interference of LTD exists. If the threshold $\theta(t)$ is set between 0 and $+1$, $\xi_i^{t+1} = 0$ isn't influenced by the interference of $\xi_i^{t-1} = 1$. When $\xi_i^{t+1} = 1$ and $\xi_i^{t-1} = 1$, the interference *does* influence the retrieval of $\boldsymbol{\xi}^{t+1}$. We consider the probability distribution of the internal potential $u_i(t)$ to examine how the interference of LTD influences the retrieval of $\boldsymbol{\xi}^{t+1}$. The probability of $\xi_i^{t+1} = 1$ and $\xi_i^{t-1} = 1$ is $f^2$, that of $\xi_i^{t+1} = 1$ and $\xi_i^{t-1} = 0$ is $f - f^2$, that of $\xi_i^{t+1} = 0$ and $\xi_i^{t-1} = 1$ is $f - f^2$, and that of $\xi_i^{t+1} = 0$ and $\xi_i^{t-1} = 0$ is $(1-f)^2$. Then the probability distribution of $u_i(t)$ is given by this equation

$$\mathrm{Prob}(u_i(t)) = (f-f^2)\delta(u_i(t)-1)+(1-2f+2f^2)\delta(u_i(t))+(f-f^2)\delta(u_i(t)+1). \tag{8}$$

Since the threshold $\theta(t)$ is set between 0 and $+1$, the state $x_i(t+1)$ is 1 with probability $f - f^2$ and 0 with $1 - f + f^2$. The overlap between the state $\boldsymbol{x}(t+1)$ and the memory pattern $\boldsymbol{\xi}^{t+1}$ is given by,

$$m^{t+1}(t+1) = \frac{1}{Nf(1-f)}\sum_{i=1}^{N}(\xi_i^{t+1} - f)x_i(t+1) = 1 - f. \tag{9}$$

In a sparse limit, $f \to 0$, the probability of $\xi_i^{t+1} = 1$ and $\xi_i^{t-1} = 1$ approaches 0. This means that the interference of LTD disappears in a sparse limit, and the model can retrieve the next pattern $\boldsymbol{\xi}^{t+1}$. Then the overlap $m^{t+1}(t+1)$ approaches 1.

Next, we discuss whether the information on the firing rate is indispensable for TAH or not. To investigate this, we consider the case that the number of memory patterns is extensively large, i.e., $p \sim O(N)$. Using the equation (9), the internal potential $u_i(t)$ of the $i$-th neuron at time $t$ is represented as,

$$u_i(t) = (\xi_i^{t+1} - \xi_i^{t-1})m^t(t) + z_i(t), \tag{10}$$

$$z_i(t) = \sum_{\mu \neq t}^{p}(\xi_i^{\mu+1} - \xi_i^{\mu-1})m^\mu(t). \tag{11}$$

$z_i(t)$ is called the "cross-talk noise", which represents contributions from non-target patterns excluding $\boldsymbol{\xi}^{t-1}$ and prevents the target pattern $\boldsymbol{\xi}^{t+1}$ from being retrieved. This disappeared in the finite loading case, $p \sim O(1)$.

It is well-known that the covariance learning is indispensable when the sparse patterns are embedded in a network as attractors [15, 16]. Under sparse coding schemes,

unless the covariance learning is employed, the cross-talk noise *does* diverge in the large $N$ limit. Consequently, the patterns can not be stored. The information on the firing rate for the stored patterns is *not* indispensable for TAH, although it is indispensable for the covariance learning. We use the method of the "statistical neurodynamics" [17, 18] to examine whether the variance of cross-talk noise diverges or not. If a pattern sequence can be stored, the cross-talk noise is obeyed by a Gaussian distribution with mean 0 and time-dependent variance $\sigma^2(t)$. Otherwise, $\sigma^2(t)$ diverges. Since $\sigma^2(t)$ is changing over time, it is necessary to control a threshold at an appropriate value at each time step [15, 16]. According to the statistical neurodynamics, we obtain the recursive equations for the overlap $m^t(t)$ between the network state $\boldsymbol{x}(t)$ and the target pattern $\boldsymbol{\xi}^t$ and the variance $\sigma^2(t)$. The details of the derivation will be shown elsewhere. Here, we show the recursive equations for $m^t(t)$ and $\sigma^2(t)$,

$$m^t(t) = \frac{1-2f}{2}\mathrm{erf}(\phi_0) - \frac{1-f}{2}\mathrm{erf}(\phi_1) + \frac{f}{2}\mathrm{erf}(\phi_2), \tag{12}$$

$$\sigma^2(t) = \sum_{a=0}^{t} {}_{2(a+1)}\mathrm{C}_{(a+1)}\alpha q(t-a) \prod_{b=1}^{a} U^2(t-b+1), \tag{13}$$

$$U(t) = \frac{1}{\sqrt{2\pi}\sigma(t-1)}\{(1-2f+2f^2)e^{-\phi_0^2} + f(1-f)(e^{-\phi_1^2} + e^{-\phi_2^2})\}, \tag{14}$$

$$q(t) = \frac{1}{2}\left(1 - (1-2f+2f^2)\mathrm{erf}(\phi_0) - f(1-f)(\mathrm{erf}(\phi_1) + \mathrm{erf}(\phi_2))\right), \tag{15}$$

$$\mathrm{erf}(y) = \frac{2}{\sqrt{\pi}}\int_0^y \exp(-u^2)du, \quad {}_b\mathrm{C}_a = \frac{b!}{a!(b-a)!}, \quad a! = a \times (a-1) \times \cdots \times 1,$$

$$\phi_0 = \frac{\theta(t-1)}{\sqrt{2}\sigma(t-1)}, \phi_1 = \frac{-m^{t-1}(t-1) + \theta(t-1)}{\sqrt{2}\sigma(t-1)}, \phi_2 = \frac{m^{t-1}(t-1) + \theta(t-1)}{\sqrt{2}\sigma(t-1)}.$$

These equations reveal that the variance $\sigma^2(t)$ of cross-talk noise does *not* diverge as long as a pattern sequence can be retrieved. This result means that TAH qualitatively has the same effect as the covariance learning.

Next, we discuss the mechanism that the variance of cross-talk noise does not diverge. Let us consider the equation (5). Synaptic weight $J_{ij}$ from $j$-th neuron to $i$-th neuron is also derived as follows,

$$J_{ij} = \frac{1}{Nf(1-f)}\sum_{\mu=1}^{p}(\xi_i^{\mu+1}\xi_j^\mu - \xi_i^{\mu-1}\xi_j^\mu) = \frac{1}{Nf(1-f)}\sum_{\mu=1}^{p}(\xi_i^\mu\xi_j^{\mu-1} - \xi_i^\mu\xi_j^{\mu+1})$$

$$= \frac{1}{Nf(1-f)}\sum_{\mu=1}^{p}\xi_i^\mu\left\{(\xi_j^{\mu-1} - f) - (\xi_j^{\mu+1} - f)\right\} \tag{16}$$

This equation implies that TAH has the information on the firing rate of the memory patterns when spatio-temporal patterns are embedded in a network. Therefore, the variance of cross-talk noise doesn't diverge, and this is another factor for the network learned by TAH to store and retrieve a pattern sequence. We conclude that the difference in spike times induces LTP or LTD, and the effect of the firing rate information can be canceled out by this spike times difference.

## 4    Results

We investigate the property of our model and examine the following two conditions: a fixed threshold and a time-dependent threshold, using the statistical neurodynamics and computer simulations.

Figure 2 shows how the overlap $m^t(t)$ and the mean firing rate of the network, $\bar{x}(t) = \frac{1}{N}\sum_i x_i(t)$, depend on the loading rate $\alpha$ when the mean firing rate of the memory pattern is $f = 0.1$ and the threshold is $\theta = 0.52$, where the storage capacity is maximum with respect to the threshold $\theta$. The stored pattern sequence can be retrieved when the initial overlap $m^1(1)$ is greater than the critical value $m_C$. The lower line indicates how the critical initial overlap $m_C$ depends on the loading rate $\alpha$. In other words, the lower line represents the basin of attraction for the retrieved sequence. The upper line denotes a steady value of overlap $m^t(t)$ when the pattern sequence is retrieved. $m^t(t)$ is obtained by setting the initial state to the first memory pattern: $\boldsymbol{x}(1) = \boldsymbol{\xi}^1$. In this case, the storage capacity is $\alpha_C = 0.27$. The dashed line shows a steady value of the normalized mean firing rate of network, $\bar{x}(t)/f$, for the pattern sequence. The data points and error bars indicate the results of the computer simulations with 5000 neurons: $N = 5000$. The former indicates mean values and the latter does variances in 10 trials. Since the results

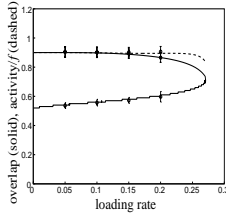

Figure 2 ! ! The critical overlap (the lower line) and the overlap at the stationary state (the upper line). The dashed line shows the mean firing rate of the network divided firing rate which is 0.1. The threshold is 0.52 and the number of neurons is 5000. The data points and error bars show the means and variances, respectively, in 10 trials of computer simulations. The storage capacity is 0.27.

of the computer simulations coincide with those of the statistical neurodynamics, hereafter, we show the results only of the statistical neurodynamics.

Next, we examine the threshold control scheme in the equation (6), where the threshold is controlled to maintain the mean firing rate of the network at $f$. $q(t)$ in equation (15) is equal to the mean firing rate because $q(t) = \frac{1}{N}\sum_{i=1}^{N}(x_i(t))^2 = \frac{1}{N}\sum_{i=1}^{N} x_i(t)$ under the condition $x_i(t) = \{0,1\}$. Thus, the threshold is adjusted to satisfy the following equation,

$$f = q(t) = \frac{1}{2}\left(1 - (1 - 2f + 2f^2)\mathrm{erf}(\phi_0) - f(1-f)(\mathrm{erf}(\phi_1) + \mathrm{erf}(\phi_2))\right). \quad (17)$$

Figure 3 shows the overlap $m^t(t)$ as a function of loading rate $\alpha$ with $f = 0.1$. The storage capacity is $\alpha_C = 0.234$. The basin of attraction becomes larger than that of the fixed threshold condition, $\theta = 0.52$ (Figure 2). Thus, the network becomes robust against noise. This means that even if the initial state $\boldsymbol{x}(1)$ is different from the first memory pattern $\boldsymbol{\xi}^1$, that is, the state includes a lot of noise, the pattern sequence can be retrieved.

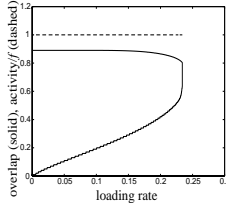

Figure 3 ! ! The critical overlap (the lower line) and the overlap at the stationary state (the upper line) when the threshold is changing over time to maintain mean firing rate of the network at $f$. The dashed line shows the mean firing rates of the network divided firing rate which is 0.1. The basin of attraction become larger than that of the fixed threshold condition: Figure 2.

Finally, we discuss how the storage capacity depends on the firing rate $f$ of the memory pattern. It is known that the storage capacity diverges as $\frac{1}{f|\log f|}$ in a sparse limit, $f \to 0$ [19, 20]. Therefore, we investigate the asymptotic property

of the storage capacity in a sparse limit. Figure 4 shows how the storage capacity depends on the firing rate where the threshold is controlled to maintain the network activity at $f$ (symbol $\circ$). The storage capacity diverges as $\frac{1}{f|\log f|}$ in a sparse limit.

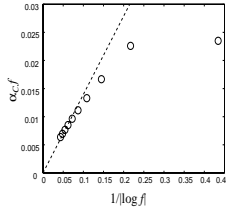

Figure 4 ! ! The storage capacity as a function of $f$ in the case of maintaining activity at $f$ (symbol $\circ$). Ths storage capacity diverges as $\frac{1}{f|\log f|}$ in a sparse limit.

## 5  Discussion

Using a simple neural network model, we have discussed the mechanism that TAH enables the network to store and retrieve a pattern sequence. First, we showed that the interference of LTP and LTD disappeared in a sparse coding scheme. This is a factor to enable the network to store and retrieve a pattern sequence. Next, we showed the mechanism that TAH qualitatively had the same effect as the covariance learning by analyzing the stability of the stored pattern sequence and the retrieval process by means of the statistical neurodynamics. Consequently, the variance of cross-talk noise didn't diverge, and this is another factor for the network learned by TAH to store and retrieve a pattern sequence. We conclude that the difference in spike times induces LTP or LTD, and the effect of the firing rate information can be canceled out by this spike times difference. We investigated the property of our model. To improve the retrieval property of the basin of attraction, we introduced a threshold control algorithm where a threshold value was adjusted to maintain the mean firing rate of the network at that of a memory pattern. As a result, we found that this scheme enlarged the basin of attraction, and that the network became robust against noise. We also found that the loading rate diverged as $\frac{1}{f|\log f|}$ in a sparse limit, $f \to 0$.

Here, we compare the storage capacity of our model with that of the model using the covariance learning (Figure 5). The dynamical equations of the model using the covariance learning is derived by Kitano and Aoyagi [13]. We calculate the storage capacity $\alpha_C^{COV}$ from their dynamical equations and compare these of our model, $\alpha_C^{TAH}$, by the ratio of $\alpha_C^{TAH}/\alpha_C^{COV}$. The threshold control method is the same as in this paper. As $f$ decreases, the ratio of storage capacities approaches 0.5. The contribution of LTD reduces the storage capacity of our model to half. Therefore, in terms of the storage capacity, the covariance learning is better than TAH. But, as we discussed previously, the information of the firing rate is indispensable in TAH. In biological systems, to get the information of the firing rate is difficult.

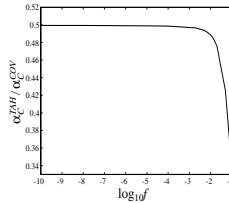

Figure 5 ! ! The comparison of the storage capacity of our model with that of the model using the covariance learning. As $f$ decreases, the ratio of storage capacity approaches 0.5.

## References

[1] G. Q. Bi and M. M. Poo. Synaptic modifications in cultured hippocampal neurons: Dependence on spike timing, synaptic strength, and postsynaptic cell

type. *The Journal of Neuroscience*, 18:10464–10472, 1998.

[2] H. Markram, J. Lübke, M. Frotscher, and B. Sakmann. Regulation of synaptic efficacy by coincidence of postsynaptic aps and epsps. *Science*, 275:213–215, 1997.

[3] L. I. Zhang, H. W. Tao, C. E. Holt, W. A. Harris, and M. M. Poo. A critical window for cooperation and competition among developing retinotectal synapses. *Nature*, 395:37–44, 1998.

[4] L. F. Abbott and S. Song. Temporally asymmetric hebbian learning, spike timing and neuronal response variability. In *Advances in Neural Information Processing Systems 11*, pages 69–75. MIT Press, 1999.

[5] J. Rubin, D. D. Lee, and H. Sompolinsky. Equilibrium properties of temporally asymmetric hebbian plasticity. *Physical Review Letters*, 86:364–367, 2001.

[6] S. Song, K. D. Miller, and L. F. Abbott. Competitive hebbian learning through spike-timing-dependent synaptic plasticity. *Nature Neuroscience*, 3:919–926, 2000.

[7] W. Gerstner, R. Kempter, J. L. van Hemmen, and H. Wagner. A neuronal learning rule for sub-millisecond temporal coding. *Nature*, 383:76–78, 1996.

[8] R. Kempter, W. Gerstner, and J. L. van Hemmen. Hebbian learning and spiking neurons. *Physical Review E*, 59:4498–4514, 1999.

[9] P. Munro and G. Hernandez. LTD facilitates learning in a noisy environment. In *Advances in Neural Information Processing Systems 12*, pages 150–156. MIT Press, 2000.

[10] R. P. N. Rao and T. J. Sejnowski. Predictive sequence learning in recurrent neocortical circuits. In *Advances in Neural Information Processing Systems 12*, pages 164–170. MIT Press, 2000.

[11] M. Yoshioka. to be published in *Physical Review E*, 2001.

[12] G. Chechik, I. Meilijson, and E. Ruppin. Effective learning requires neuronal remodeling of hebbian synapses. In *Advances in Neural Information Processing Systems 11*, pages 96–102. MIT Press, 1999.

[13] K. Kitano and T. Aoyagi. Retrieval dynamics of neural networks for sparsely coded sequential patterns. *Journal of Physics A: Mathematical and General*, 31:L613–L620, 1998.

[14] M. Miyashita. Neuronal correlate of visual associative long-term memory in the primate temporal cortex. *Nature*, 335:817–820, 1988.

[15] S. Amari. Characteristics of sparsely encoded associative memory. *Neural Networks*, 2:1007–1018, 1989.

[16] M. Okada. Notions of associative memory and sparse coding. *Neural Networks*, 9:1429–1458, 1996.

[17] S. Amari and K. Maginu. Statistical neurodynamics of various versions of correlation associative memory. *Neural Networks*, 1:63–73, 1988.

[18] M. Okada. A hierarchy of macrodynamical equations for associative memory. *Neural Networks*, 8:833–838, 1995.

[19] M. V. Tsodyks and M. V. Feigle'man. The enhanced strage capacity in neural networks with low activity level. *Europhysics Letters*, 6:101–105, 1988.

[20] C. J. Perez-Vicente and D. J. Amit. Optimized network for sparsely coded patterns. *Journal of Physics A: Mathematical and General*, 22:559–569, 1989.
